# Resolution Limits of Sparse Coding in High Dimensions[*]

Alyson K. Fletcher,[†] Sundeep Rangan,[‡] and Vivek K Goyal[§]

## Abstract

This paper addresses the problem of sparsity pattern detection for unknown $k$-sparse $n$-dimensional signals observed through $m$ noisy, random linear measurements. Sparsity pattern recovery arises in a number of settings including statistical model selection, pattern detection, and image acquisition. The main results in this paper are necessary and sufficient conditions for asymptotically-reliable sparsity pattern recovery in terms of the dimensions $m$, $n$ and $k$ as well as the signal-to-noise ratio (SNR) and the minimum-to-average ratio (MAR) of the nonzero entries of the signal. We show that $m > 2k \log(n-k)/(\mathsf{SNR} \cdot \mathsf{MAR})$ is necessary for any algorithm to succeed, regardless of complexity; this matches a previous sufficient condition for maximum likelihood estimation within a constant factor under certain scalings of $k$, SNR and MAR with $n$. We also show a sufficient condition for a computationally-trivial thresholding algorithm that is larger than the previous expression by only a factor of $4(1 + \mathsf{SNR})$ and larger than the requirement for lasso by only a factor of $4/\mathsf{MAR}$. This provides insight on the precise value and limitations of convex programming-based algorithms.

## 1 Introduction

Sparse signal models have been used successfully in a variety of applications including wavelet-based image processing and pattern recognition. Recent research has shown that certain naturally-occurring neurological processes may exploit sparsity as well [1–3]. For example, there is now evidence that the V1 visual cortex naturally generates a sparse representation of the visual data relative to a certain Gabor-like basis. Due to the nonlinear nature of sparse signal models, developing and analyzing algorithms for sparse signal processing has been a major research challenge.

This paper considers the problem of estimating sparse signals in the presence of noise. We are specifically concerned with understanding the theoretical estimation limits and how far practical algorithms are from those limits. In the context of visual cortex modeling, this analysis may help us understand what visual features are resolvable from visual data. To keep the analysis general, we consider the following abstract estimation problem: An unknown sparse signal $x$ is modeled as an $n$-dimensional real vector with $k$ nonzero components. The locations of the nonzero components is called the *sparsity pattern*. We consider the problem of detecting the sparsity pattern of $x$ from an $m$-dimensional measurement vector $y = Ax + d$, where $A \in \mathbb{R}^{m \times n}$ is a known measurement matrix and $d \in \mathbb{R}^m$ is an additive noise vector with a known distribution. We are interested in

[*]This work was supported in part by a University of California President's Postdoctoral Fellowship, NSF CAREER Grant CCF-643836, and the Centre Bernoulli at École Polytechnique Fédérale de Lausanne.

[†]A. K. Fletcher (email: alyson@eecs.berkeley.edu) is with the Department of Electrical Engineering and Computer Sciences, University of California, Berkeley.

[‡]S. Rangan (email: srangan@qualcomm.com) is with Qualcomm Technologies, Bedminster, NJ.

[§]V. K. Goyal (email: vgoyal@mit.edu) is with the Department of Electrical Engineering and Computer Science and the Research Laboratory of Electronics, Massachusetts Institute of Technology.

| | finite SNR | SNR $\to \infty$ |
|---|---|---|
| Any algorithm must fail | $m < \frac{2}{\text{MAR}\cdot\text{SNR}} k \log(n-k) + k - 1$ <br> Theorem 1 | $m \leq k$ <br> (elementary) |
| Necessary and sufficient for lasso | unknown (expressions above and right are necessary) | $m \asymp 2k \log(n-k) + k + 1$ <br> Wainwright [14] |
| Sufficient for thresholding estimator (11) | $m > \frac{8(1+\text{SNR})}{\text{MAR}\cdot\text{SNR}} k \log(n-k)$ <br> Theorem 2 | $m > \frac{8}{\text{MAR}} k \log(n-k)$ <br> from Theorem 2 |

Table 1: Summary of Results on Measurement Scaling for Reliable Sparsity Recovery
(see body for definitions and technical limitations)

determining necessary and sufficient conditions on the ability to reliably detect the sparsity pattern based on problem dimensions $m$, $n$ and $k$, and signal and noise statistics.

**Previous work.** While optimal sparsity pattern detection is NP-hard [4], greedy heuristics (matching pursuit [5] and its variants) and convex relaxations (basis pursuit [6], lasso [7], and others) have been widely-used since at least the mid 1990s. While these algorithms worked well in practice, until recently, little could be shown analytically about their performance. Some remarkable recent results are sets of conditions that can guarantee exact sparsity recovery based on certain simple "incoherence" conditions on the measurement matrix $A$ [8–10].

These conditions and others have been exploited in developing the area of "compressed sensing," which considers large random matrices $A$ with i.i.d. components [11–13]. The main theoretical result are conditions that guarantee sparse detection with convex programming methods. The best of these results is due to Wainwright [14], who shows that the scaling

$$m \asymp 2k \log(n-k) + k + 1. \tag{1}$$

is necessary and sufficient for lasso to detect the sparsity pattern when $A$ has Gaussian entries, provided the SNR scales to infinity.

**Preview.** This paper presents new necessary and sufficient conditions, summarized in Table 1 along with Wainwright's lasso scaling (1). The parameters MAR and SNR represent the minimum-to-average and signal-to-noise ratio, respectively. The exact definitions and measurement model are given below.

The necessary condition applies to all algorithms, regardless of complexity. Previous necessary conditions had been based on information-theoretic analyses such as [15–17]. More recent publications with necessary conditions include [18–21]. As described in Section 3, our new necessary condition is stronger than previous bounds in certain important regimes.

The sufficient condition is derived for a computationally-trivial thresholding estimator. By comparing with the lasso scaling, we argue that main benefits of more sophisticated methods, such as lasso, is not generally in the scaling with respect to $k$ and $n$ but rather in the dependence on the minimum-to-average ratio.

## 2 Problem Statement

Consider estimating a $k$-sparse vector $x \in \mathbb{R}^n$ through a vector of observations,

$$y = Ax + d, \tag{2}$$

where $A \in \mathbb{R}^{m \times n}$ is a random matrix with i.i.d. $\mathcal{N}(0, 1/m)$ entries and $d \in \mathbb{R}^m$ is i.i.d. unit-variance Gaussian noise. Denote the sparsity pattern of $x$ (positions of nonzero entries) by the set $I_{\text{true}}$, which is a $k$-element subset of the set of indices $\{1, 2, \ldots, n\}$. Estimates of the sparsity pattern will be denoted by $\hat{I}$ with subscripts indicating the type of estimator. We seek conditions under which there exists an estimator such that $\hat{I} = I_{\text{true}}$ with high probability.

In addition to the signal dimensions, $m$, $n$ and $k$, we will show that there are two variables that dictate the ability to detect the sparsity pattern reliably: the signal-to-noise ratio (SNR), and what we will call the *minimum-to-average ratio* (MAR).

The SNR is defined by

$$\mathsf{SNR} = \frac{\mathbf{E}[\|Ax\|^2]}{\mathbf{E}[\|d\|^2]} = \frac{\mathbf{E}[\|Ax\|^2]}{m}. \tag{3}$$

Since we are considering $x$ as an unknown deterministic vector, the SNR can be further simplified as follows: The entries of $A$ are i.i.d. $\mathcal{N}(0, 1/m)$, so columns $a_i \in \mathbb{R}^m$ and $a_j \in \mathbb{R}^m$ of $A$ satisfy $\mathbf{E}[a_i' a_j] = \delta_{ij}$. Therefore, the signal energy is given by

$$\mathbf{E}\left[\|Ax\|^2\right] = \sum_{i,j \in I_{\mathrm{true}}} \mathbf{E}\left[a_i' a_j x_i x_j\right] = \sum_{i,j \in I_{\mathrm{true}}} x_i x_j \delta_{ij} = \|x\|^2.$$

Substituting into the definition (3), the SNR is given by

$$\mathsf{SNR} = \frac{1}{m}\|x\|^2. \tag{4}$$

The minimum-to-average ratio of $x$ is defined as

$$\mathsf{MAR} = \frac{\min_{j \in I_{\mathrm{true}}} |x_j|^2}{\|x\|^2/k}. \tag{5}$$

Since $\|x\|^2/k$ is the average of $\{|x_j|^2 \mid j \in I_{\mathrm{true}}\}$, $\mathsf{MAR} \in (0, 1]$ with the upper limit occurring when all the nonzero entries of $x$ have the same magnitude.

One final value that will be important is the *minimum component SNR*, defined as

$$\mathsf{SNR}_{\min} = \frac{1}{\mathbf{E}\|d\|^2} \min_{j \in I_{\mathrm{true}}} \mathbf{E}\|a_j x_j\|^2 = \frac{1}{m} \min_{j \in I_{\mathrm{true}}} |x_j|^2. \tag{6}$$

The quantity $\mathsf{SNR}_{\min}$ has a natural interpretation: The numerator, $\min \mathbf{E}\|a_j x_j\|^2$, is the signal power due to the smallest nonzero component of $x$, while the denominator, $\mathbf{E}\|d\|^2$, is the total noise power. The ratio $\mathsf{SNR}_{\min}$ thus represents the contribution to the SNR from the smallest nonzero component of the unknown vector $x$. Observe that (3) and (5) show

$$\mathsf{SNR}_{\min} = \frac{1}{k}\mathsf{SNR} \cdot \mathsf{MAR}. \tag{7}$$

**Normalizations.** Other works use a variety of normalizations, e.g.: the entries of $A$ have variance $1/n$ in [13, 19]; the entries of $A$ have unit variance and the variance of $d$ is a variable $\sigma^2$ in [14, 17, 20, 21]; and our scaling of $A$ and a noise variance of $\sigma^2$ are used in [22]. This necessitates great care in comparing results.

To facilitate the comparison we have expressed all our results in terms of SNR, MAR and $\mathsf{SNR}_{\min}$ as defined above. All of these quantities are *dimensionless*, in that if either $A$ and $d$ or $x$ and $d$ are scaled together, these ratios will not change. Thus, the results can be applied to *any* scaling of $A$, $d$ and $x$, provided that the quantities SNR, MAR and $\mathsf{SNR}_{\min}$ are computed appropriately.

## 3 Necessary Condition for Sparsity Recovery

We first consider sparsity recovery without being concerned with computational complexity of the estimation algorithm. Since the vector $x \in \mathbb{R}^n$ is $k$-sparse, the vector $Ax$ belongs to one of $L = \binom{n}{k}$ subspaces spanned by $k$ of the $n$ columns of $A$. Estimation of the sparsity pattern is the selection of one of these subspaces, and since the noise $d$ is Gaussian, the probability of error is minimized by choosing the subspace closest to the observed vector $y$. This results in the maximum likelihood (ML) estimate.

Mathematically, the ML estimator can be described as follows. Given a subset $J \subseteq \{1, 2, \ldots, n\}$, let $P_J y$ denote the orthogonal projection of the vector $y$ onto the subspace spanned by the vectors $\{a_j \mid j \in J\}$. The ML estimate of the sparsity pattern is

$$\hat{I}_{\mathrm{ML}} = \arg\max_{J \,:\, |J|=k} \|P_J y\|^2,$$

where $|J|$ denotes the cardinality of $J$. That is, the ML estimate is the set of $k$ indices such that the subspace spanned by the corresponding columns of $A$ contain the maximum signal energy of $y$.

Since the number of subspaces $L$ grows exponentially in $n$ and $k$, an exhaustive search is, in general, computationally infeasible. However, the performance of ML estimation provides a lower bound on the number of measurements needed by any algorithm that cannot exploit a priori information on $x$ other than it being $k$-sparse.

ML estimation for sparsity recovery was first examined in [17]. There, it was shown that there exists a constant $C > 0$ such that the condition

$$m > C \max \left\{ \frac{\log(n-k)}{\mathsf{SNR}_{\min}}, k \log\left(\frac{n}{k}\right) \right\} = C \max \left\{ \frac{k \log(n-k)}{\mathsf{SNR} \cdot \mathsf{MAR}}, k \log\left(\frac{n}{k}\right) \right\} \tag{8}$$

is *sufficient* for ML to asymptotically reliably recover the sparsity pattern. Note that the equality between the two expressions in (8) is a consequence of (7). Our first theorem provides a corresponding necessary condition.

**Theorem 1** *Let $k = k(n)$, $m = m(n)$, $\mathsf{SNR} = \mathsf{SNR}(n)$ and $\mathsf{MAR} = \mathsf{MAR}(n)$ be deterministic sequences in $n$ such that $\lim_{n \to \infty} k(n) = \infty$ and*

$$m(n) < \frac{2-\delta}{\mathsf{SNR}_{\min}} \log(n-k) + k - 1 = \frac{2-\delta}{\mathsf{MAR} \cdot \mathsf{SNR}} k \log(n-k) + k - 1 \tag{9}$$

*for some $\delta > 0$. Then even the ML estimator asymptotically cannot detect the sparsity pattern, i.e.,*

$$\lim_{n \to \infty} \Pr\left(\hat{I}_{\mathrm{ML}} = I_{\mathrm{true}}\right) = 0.$$

*Proof sketch:* The basic idea in the proof is to consider an "incorrect" subspace formed by removing one of the $k$ correct vectors with the least energy, and adding one of the $n-k$ incorrect vectors with largest energy. The change in energy can be estimated using tail distributions of chi-squared random variables. A complete proof appears in [23].

The theorem provides a simple lower bound on the minimum number of measurements required to recover the sparsity pattern in terms of $k$, $n$ and the minimum component SNR, $\mathsf{SNR}_{\min}$. Note that the equivalence between the two expressions in (9) is due to (7).

**Remarks.**

1. The theorem strengthens an earlier necessary condition in [18] which showed that there exists a $C > 0$ such that
$$m = \frac{C}{\mathsf{SNR}} k \log(n-k)$$
   is necessary for asymptotic reliable recovery. Theorem 1 strengthens the result to reflect the dependence on MAR and make the constant explicit.

2. The theorem applies for any $k(n)$ such that $\lim_{n \to \infty} k(n) = \infty$, including both cases with $k = o(n)$ and $k = \Theta(n)$. In particular, under linear sparsity ($k = \alpha n$ for some constant $\alpha$), the theorem shows that
$$m \asymp \frac{2\alpha}{\mathsf{MAR} \cdot \mathsf{SNR}} n \log n$$
   measurements are necessary for sparsity recovery. Similarly, if $m/n$ is bounded above by a constant, then sparsity recovery will certainly fail unless
$$k = O\left(\mathsf{SNR} \cdot \mathsf{MAR} \cdot n / \log n\right).$$
   In particular, when $\mathsf{SNR} \cdot \mathsf{MAR}$ is bounded, the sparsity ratio $k/n$ must approach zero.

3. In the case where $\mathsf{SNR} \cdot \mathsf{MAR}$ and the sparsity ratio $k/n$ are both constant, the sufficient condition (8) reduces to
$$m = (C/(\mathsf{SNR} \cdot \mathsf{MAR}))k \log(n-k),$$
   which matches the necessary condition in (9) within a constant factor.

4. In the case of $\mathsf{MAR} \cdot \mathsf{SNR} < 1$, the bound (9) improves upon the necessary condition of [14] for the asymptotic success of lasso by the factor $(\mathsf{MAR} \cdot \mathsf{SNR})^{-1}$.

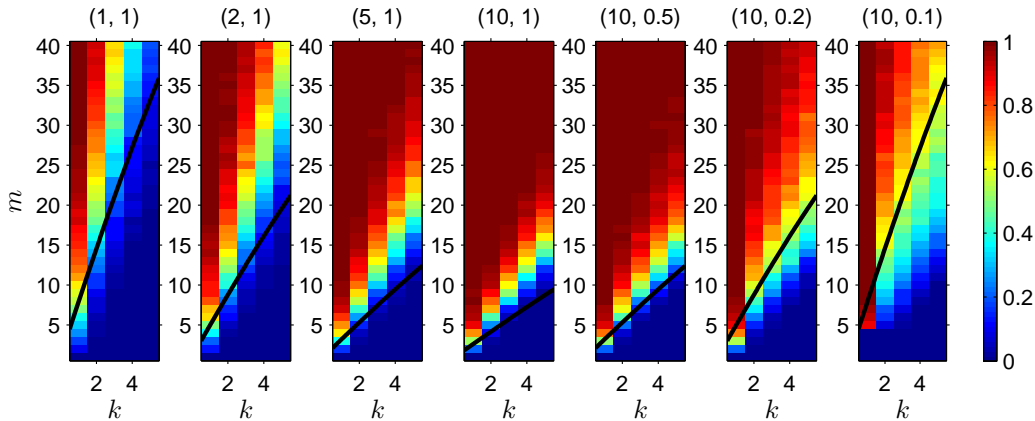

Figure 1: Simulated success probability of ML detection for $n = 20$ and many values of $k$, $m$, SNR, and MAR. Each subfigure gives simulation results for $k \in \{1, 2, \ldots, 5\}$ and $m \in \{1, 2, \ldots, 40\}$ for one (SNR, MAR) pair. Each subfigure heading gives (SNR, MAR). Each point represents at least 500 independent trials. Overlaid on the color-intensity plots is a black curve representing (9).

5. The bound (9) can be compared against information-theoretic bounds such as those in [15–17, 20, 21]. For example, a simple capacity argument in [15] shows that

$$m \geq \frac{2 \log_2 \binom{n}{k}}{\log_2(1 + \mathsf{SNR})} \tag{10}$$

is necessary. When the sparsity ratio $k/n$ and SNR are both fixed, $m$ can satisfy (10) while growing only linearly with $k$. In contrast, Theorem 1 shows that with sparsity ratio and SNR $\cdot$ MAR fixed, $m = \Omega(k \log(n-k))$ is necessary for reliable sparsity recovery. That is, the number of measurements must grow *superlinearly* in $k$ in the linear sparsity regime with bounded SNR. In the sublinear regime where $k = o(n)$, the capacity-based bound (10) may be stronger than (9) depending on the scaling of SNR, MAR and other terms.

6. Results more similar to Theorem 1—based on direct analyses of error events rather than information-theoretic arguments—appeared in [18, 19]. The previous results showed that with fixed SNR as defined here, sparsity recovery with $m = \Theta(k)$ must fail. The more refined analysis in this paper gives the additional $\log(n - k)$ factor and the precise dependence on MAR $\cdot$ SNR.

7. Theorem 1 is not contradicted by the relevant sufficient condition of [20, 21]. That sufficient condition holds for scaling that gives linear sparsity and MAR $\cdot$ SNR $= \Omega(\sqrt{n \log n})$. For MAR $\cdot$ SNR $= \sqrt{n \log n}$, Theorem 1 shows that fewer than $m \asymp 2\sqrt{k \log k}$ measurements will cause ML decoding to fail, while [21, Thm. 3.1] shows that a typicality-based decoder will succeed with $m = \Theta(k)$ measurements.

8. The necessary condition (9) shows a dependence on the minimum-to-average ratio MAR instead of just the average power through SNR. Thus, the bound shows the negative effects of relatively small components. Note that [17, Thm. 2] appears to have dependence on the minimum power as well, but is actually only proven for the case MAR $= 1$.

**Numerical validation.** Computational confirmation of Theorem 1 is technically impossible, and even qualitative support is hard to obtain because of the high complexity of ML detection. Nevertheless, we may obtain some evidence through Monte Carlo simulation.

Fig. 1 shows the probability of success of ML detection for $n = 20$ as $k$, $m$, SNR, and MAR are varied. Signals with MAR $< 1$ are created by having one small nonzero component and $k - 1$ equal, larger nonzero components. Taking any one column of one subpanel from bottom to top shows that as $m$ is increased, there is a transition from ML failing to ML succeeding. One can see that (9) follows the failure-success transition qualitatively. In particular, the empirical dependence on SNR and MAR approximately follows (9). Empirically, for the (small) value of $n = 20$, it seems that with MAR $\cdot$ SNR held fixed, sparsity recovery becomes easier as SNR increases (and MAR decreases).

## 4 Sufficient Condition for Thresholding

Consider the following simple estimator. As before, let $a_j$ be the $j$th column of the random matrix $A$. Define the *thresholding estimate* as

$$\hat{I}_{\text{thresh}} = \left\{ j \ : \ |a_j'y|^2 > \mu \right\}, \tag{11}$$

where $\mu > 0$ represents a threshold level. This algorithm simply correlates the observed signal $y$ with all the frame vectors $a_j$ and selects the indices $j$ where the correlation energy exceeds a certain level $\mu$. It is significantly simpler than both lasso and matching pursuit and is not meant to be proposed as a competitive alternative. Rather, we consider thresholding simply to illustrate what precise benefits lasso and more sophisticated methods bring.

Sparsity pattern recovery by thresholding was studied in [24], which proves a sufficient condition when there is no noise. The following theorem improves and generalizes the result to the noisy case.

**Theorem 2** *Let $k = k(n)$, $m = m(n)$, SNR $=$ SNR$(n)$ and MAR $=$ MAR$(n)$ be deterministic sequences in $n$ such that $\lim_{n \to \infty} k = \infty$ and*

$$m > \frac{8(1 + \delta)(1 + \mathsf{SNR})}{\mathsf{MAR} \cdot \mathsf{SNR}} k \log(n - k) \tag{12}$$

*for some $\delta > 0$. Then, there exists a sequence of threshold levels $\mu = \mu(n)$, such that thresholding asymptotically detects the sparsity pattern, i.e.,*

$$\lim_{n \to \infty} \Pr \left( \hat{I}_{\text{thresh}} = I_{\text{true}} \right) = 1.$$

*Proof sketch:* Using tail distributions of chi-squared random variables, it is shown that the minimum value for the correlation $|a_j'y|^2$ when $j \in I_{\text{true}}$ is greater than the maximum correlation when $j \notin I_{\text{true}}$. A complete proof appears in [23].

**Remarks.**

1. Comparing (9) and (12), we see that thresholding requires a factor of at most $4(1 + \mathsf{SNR})$ more measurements than ML estimation. Thus, for a fixed SNR, the optimal scaling not only does not require ML estimation, it does not even require lasso or matching pursuit—it can be achieved with a remarkably simply method.

2. Nevertheless, the gap between thresholding and ML of $4(1+\mathsf{SNR})$ measurements can be large. This is most apparent in the regime where the SNR $\to \infty$. For ML estimation, the lower bound on the number of measurements required by ML decreases to $k - 1$ as SNR $\to \infty$.[1] In contrast, with thresholding, increasing the SNR has diminishing returns: as SNR $\to \infty$, the bound on the number of measurements in (12) approaches

$$m > \frac{8}{\mathsf{MAR}} k \log(n - k). \tag{13}$$

   Thus, even with SNR $\to \infty$, the minimum number of measurements is not improved from $m = \Omega(k \log(n - k))$.

   This diminishing returns for improved SNR exhibited by thresholding is also a problem for more sophisticated methods such as lasso. For example, as discussed earlier, the analysis of [14] shows that when SNR $\cdot$ MAR $\to \infty$, lasso requires

$$m > 2k \log(n - k) + k + 1 \tag{14}$$

   for reliable recovery. Therefore, like thresholding, lasso does not achieve a scaling better than $m = O(k \log(n - k))$, even at infinite SNR.

3. There is also a gap between thresholding and lasso. Comparing (13) and (14), we see that, at high SNR, thresholding requires a factor of up to $4/\mathsf{MAR}$ more measurements than lasso. This factor is largest when MAR is small, which occurs when there are relatively small nonzero components. Thus, in the high SNR regime, the main benefit of lasso is its ability to detect small coefficients, even when they are much below the average power. However, if the range of component magnitudes is not large, so MAR is close to one, lasso and thresholding have equal performance within a constant factor.

4. The high SNR limit (13) matches the sufficient condition in [24] for the noise free case, except that the constant in (13) is tighter.

**Numerical validation.** Thresholding is extremely simple and can thus be simulated easily for large problem sizes. The results of a large number of Monte Carlo simulations are presented in [23], which also reports additional simulations of maximum likelihood estimation. With $n = 100$, the sufficient condition predicted by (12) matches well to the parameter combinations where the probability of success drops below about 0.995.

## 5  Conclusions

We have considered the problem of detecting the sparsity pattern of a sparse vector from noisy random linear measurements. Necessary and sufficient scaling laws for the number of measurements to recover the sparsity pattern for different detection algorithms were derived. The analysis reveals the effect of two key factors: the total signal-to-noise ratio (SNR), as well as the minimum-to-average ratio (MAR), which is a measure of the spread of component magnitudes. The product of these factors is $k$ times the SNR contribution from the smallest nonzero component; this product often appears.

Our main conclusions are:

- *Tight scaling laws for constant SNR and MAR.* In the regime where $\mathsf{SNR} = \Theta(1)$ and $\mathsf{MAR} = \Theta(1)$, our results show that the scaling of the number of measurements

$$m = O(k \log(n - k))$$

  is both necessary and sufficient for asymptotically reliable sparsity pattern detection. Moreover, the scaling can be achieved with a thresholding algorithm, which is computationally simpler than even lasso or basis pursuit. Under the additional assumption of linear sparsity ($k/n$ fixed), this scaling is a larger number of measurements than predicted by previous information-theoretic bounds.
- *Dependence on SNR.* While the number of measurements required for exhaustive ML estimation and simple thresholding have the same dependence on $n$ and $k$ with the SNR fixed, the dependence on SNR differs significantly. Specifically, thresholding requires a factor of up to $4(1 + \mathsf{SNR})$ more measurements than ML. Moreover, as $\mathsf{SNR} \to \infty$, the number of measurements required by ML may be as low as $m = k + 1$. In contrast, even letting $\mathsf{SNR} \to \infty$, thresholding and lasso still require $m = O(k \log(n - k))$ measurements.
- *Lasso and dependence on MAR.* Thresholding can also be compared to lasso, at least in the high SNR regime. There is a potential gap between thresholding and lasso, but the gap is smaller than the gap to ML. Specifically, in the high SNR regime, thresholding requires at most $4/\mathsf{MAR}$ more measurements than lasso. Thus, the benefit of lasso over simple thresholding is its ability to detect the sparsity pattern even in the presence of relatively small nonzero coefficients (i.e. low MAR). However, when the components of the unknown vector have similar magnitudes (MAR close to one), the gap between lasso and simple thresholding is reduced.

While our results provide both necessary and sufficient scaling laws, there is clearly a gap in terms of the scaling with the SNR. We have seen that full ML estimation could potentially have a scaling in SNR as small as $m = O(1/\mathsf{SNR}) + k - 1$. An open question is whether there is any practical algorithm that can achieve a similar scaling.

A second open issue is to determine conditions for partial sparsity recovery. The above results define conditions for recovering all the positions in the sparsity pattern. However, in many practical applications, obtaining some large fraction of these positions would be sufficient. Neither the limits of partial sparsity recovery nor the performance of practical algorithms are completely understood, though some results have been reported in [19–21, 25].

## Footnotes

[1] Of course, at least $k + 1$ measurements are necessary.

# References

[1] M. Lewicki. Efficient coding of natural sounds. *Nature Neuroscience*, 5:356–363, 2002.

[2] B. A. Olshausen and D. J. Field. Sparse coding of sensory inputs. *Curr. Op. in Neurobiology*, 14:481–487, 2004.

[3] C. J. Rozell, D. H. Johnson, R. G. Baraniuk, and B. A. Olshausen. Sparse coding via thresholding and local competition in neural circuits. *Neural Computation*, 2008. In press.

[4] B. K. Natarajan. Sparse approximate solutions to linear systems. *SIAM J. Computing*, 24(2):227–234, April 1995.

[5] S. G. Mallat and Z. Zhang. Matching pursuits with time-frequency dictionaries. *IEEE Trans. Signal Process.*, 41(12):3397–3415, Dec. 1993.

[6] S. S. Chen, D. L. Donoho, and M. A. Saunders. Atomic decomposition by basis pursuit. *SIAM J. Sci. Comp.*, 20(1):33–61, 1999.

[7] R. Tibshirani. Regression shrinkage and selection via the lasso. *J. Royal Stat. Soc., Ser. B*, 58(1):267–288, 1996.

[8] D. L. Donoho, M. Elad, and V. N. Temlyakov. Stable recovery of sparse overcomplete representations in the presence of noise. *IEEE Trans. Inform. Theory*, 52(1):6–18, Jan. 2006.

[9] J. A. Tropp. Greed is good: Algorithmic results for sparse approximation. *IEEE Trans. Inform. Theory*, 50(10):2231–2242, Oct. 2004.

[10] J. A. Tropp. Just relax: Convex programming methods for identifying sparse signals in noise. *IEEE Trans. Inform. Theory*, 52(3):1030–1051, March 2006.

[11] E. J. Candès, J. Romberg, and T. Tao. Robust uncertainty principles: Exact signal reconstruction from highly incomplete frequency information. *IEEE Trans. Inform. Theory*, 52(2):489–509, Feb. 2006.

[12] D. L. Donoho. Compressed sensing. *IEEE Trans. Inform. Theory*, 52(4):1289–1306, April 2006.

[13] E. J. Candès and T. Tao. Near-optimal signal recovery from random projections: Universal encoding strategies? *IEEE Trans. Inform. Theory*, 52(12):5406–5425, Dec. 2006.

[14] M. J. Wainwright. Sharp thresholds for high-dimensional and noisy recovery of sparsity. arXiv:0605.740v1 [math.ST]., May 2006.

[15] S. Sarvotham, D. Baron, and R. G. Baraniuk. Measurements vs. bits: Compressed sensing meets information theory. In *Proc. 44th Ann. Allerton Conf. on Commun., Control and Comp.*, Monticello, IL, Sept. 2006.

[16] A. K. Fletcher, S. Rangan, and V. K. Goyal. Rate-distortion bounds for sparse approximation. In *IEEE Statist. Sig. Process. Workshop*, pages 254–258, Madison, WI, Aug. 2007.

[17] M. J. Wainwright. Information-theoretic limits on sparsity recovery in the high-dimensional and noisy setting. Tech. Report 725, Univ. of California, Berkeley, Dept. of Stat., Jan. 2007.

[18] V. K. Goyal, A. K. Fletcher, and S. Rangan. Compressive sampling and lossy compression. *IEEE Sig. Process. Mag.*, 25(2):48–56, March 2008.

[19] G. Reeves. Sparse signal sampling using noisy linear projections. Tech. Report UCB/EECS-2008-3, Univ. of California, Berkeley, Dept. of Elec. Eng. and Comp. Sci., Jan. 2008.

[20] M. Akçakaya and V. Tarokh. Shannon theoretic limits on noisy compressive sampling. arXiv:0711.0366v1 [cs.IT]., Nov. 2007.

[21] M. Akçakaya and V. Tarokh. Noisy compressive sampling limits in linear and sublinear regimes. In *Proc. Conf. on Inform. Sci. & Sys.*, Princeton, NJ, March 2008.

[22] J. Haupt and R. Nowak. Signal reconstruction from noisy random projections. *IEEE Trans. Inform. Theory*, 52(9):4036–4048, Sept. 2006.

[23] A. K. Fletcher, S. Rangan, and V. K. Goyal. Necessary and sufficient conditions on sparsity pattern recovery. arXiv:0804.1839v1 [cs.IT]., April 2008.

[24] H. Rauhut, K. Schnass, and P. Vandergheynst. Compressed sensing and redundant dictionaries. *IEEE Trans. Inform. Theory*, 54(5):2210–2219, May 2008.

[25] S. Aeron, M. Zhao, and V. Saligrama. On sensing capacity of sensor networks for the class of linear observation, fixed SNR models. arXiv:0704.3434v3 [cs.IT]., June 2007.
